# Repeated Games against Budgeted Adversaries

**Jacob Abernethy**[*]
Division of Computer Science
UC Berkeley
jake@cs.berkeley.edu

**Manfred K. Warmuth**[†]
Department of Computer Science
UC Santa Cruz
manfred@cse.ucsc.edu

## Abstract

We study repeated zero-sum games against an adversary on a budget. Given that an adversary has some constraint on the sequence of actions that he plays, we consider what ought to be the player's best mixed strategy with knowledge of this budget. We show that, for a general class of normal-form games, the minimax strategy is indeed efficiently computable and relies on a "random playout" technique. We give three diverse applications of this new algorithmic template: a cost-sensitive "Hedge" setting, a particular problem in Metrical Task Systems, and the design of combinatorial prediction markets.

## 1 Introduction

How can we reasonably expect to learn given possibly adversarial data? Overcoming this obstacle has been one of the major successes of the Online Learning framework or, more generally, the so-called competitive analysis of algorithms: rather than measure an algorithm only by the cost it incurs, consider this cost *relative* to an optimal "comparator algorithm" which has knowledge of the data in advance. A classic example is the so-called "experts setting": assume we must predict a sequence of binary outcomes and we are given access to a set of *experts*, each of which reveals their own prediction for each outcome. After each round we learn the true outcome and, hence, which experts predicted correctly or incorrectly. The expert setting is based around a simple assumption, that while some experts' predictions may be adversarial, we have an a priori belief that there is at least one good expert whose predictions will be reasonably accurate. Under this relatively weak good-expert assumption, one can construct algorithms that have quite strong loss guarantees.

Another way to interpret this sequential prediction model is to treat it as a repeated two-player zero-sum game against an adversary *on a budget*; that is, the adversary's sequence of actions is restricted in that play ceases once the adversary exceeds the budget. In the experts setting, the assumption "there is a good expert" can be reinterpreted as a "nature shall not let the best expert err too frequently", perhaps more than some fixed number of times.

In the present paper, we develop a general framework for repeated game-playing against an adversary on a budget, and we provide a simple randomized strategy for the learner/player for a particular class of these games. The proposed algorithms are based on a technique, which we refer to as a "random playout", that has become a very popular heuristic for solving games with massively-large state spaces. Roughly speaking, a random playout in an extensive-form game is a way to measure the likely outcome at a given state by finishing the game randomly from this state. Random playouts, often known simply as Monte Carlo methods, have become particularly popular for solving the game of Go [5], which has led to much follow-up work for general games [12, 11]. The Budgeted Adversary game we consider also involves exponentially large state spaces, yet we achieve efficiency using these random playouts. The key result of this paper is that the proposed random playout is not simply a good heuristic, it is indeed *minimax optimal* for the games we consider.

---

[*]Supported by a Yahoo! PhD Fellowship and NSF grant 0830410.
[†]Supported by NSF grant IIS-0917397.

Abernethy et al [1] was the first to use a random playout strategy to optimally solve an adversarial learning problem, namely for the case of the so-called Hedge Setting introduced by Freund and Schapire [10]. Indeed, their model can be interpreted as a particular special case of a Budgeted Adversary problem. The generalized framework that we give in the first half of the paper, however, has a much larger range of applications. We give three such examples, described briefly below. More details are given in the second half of the paper.

**Cost-sensitive Hedge Setting.** In the standard Hedge setting, it is assumed that each expert suffers a cost in $[0, 1]$ on each round. But a surprisingly-overlooked case is when the cost ranges differ, where expert $i$ may suffer per-round cost in $[0, c_i]$ for some fixed $c_i > 0$. The vanilla approach, to use a generic bound of $\max_i c_i$, is extremely loose, and we know of no better bounds for this case. Our results provide the optimal strategy for this cost-sensitive Hedge setting.

**Metrical Task Systems (MTS).** The MTS problem is decision/learning problem similar to the Hedge Setting above but with an added difficulty: the learner is required to pay the cost of moving through a given metric space. Finding even a near-optimal generic algorithm has remained elusive for some time, with recent encouraging progress made in one special case [2], for the so-called "weighted-star" metric. Our results provide a simple minimax optimal algorithm for this problem.

**Combinatorial Prediction Market Design:** There has been a great deal of work in designing so-called prediction markets, where bettors may purchase contracts that pay off when the outcome of a future event is correctly guessed. One important goal of such markets is to minimize the potential risk of the "market maker" who sells the contracts and pays the winning bettors. Another goal is to design "combinatorial" markets, that is where the outcome space might be complex. The latter has proven quite challenging, and there are few positive results within this area. We show how to translate the market-design problem into a Budgeted Adversary problem, and from here how to incorporate certain kinds of combinatorial outcomes.

## 2  Preliminaries

**Notation:** We shall write $[n]$ for the set $\{1, 2, \ldots, n\}$, and $[n]^*$ to be the set of all finite-length sequences of elements of $[n]$. We will use the greek symbols $\rho$ and $\sigma$ to denote such sequences $i_1 i_2 \ldots i_T$, where $i_t \in [n]$. We let $\emptyset$ denote the empty sequence. When we have defined some $T$-length sequence $\rho = i_1 i_2 \ldots i_T$, we may write $\rho_t$ to refer to the $t$-length prefix of $\rho$, namely $\rho_t = i_1 i_2 \ldots i_t$, and clearly $t \leq T$. We will generally use $\mathbf{w}$ to refer to a distribution in $\Delta_n$, the $n$-simplex, where $w_i$ denotes the $i$th coordinate of $\mathbf{w}$. We use the symbol $\mathbf{e}_i$ to denote the $i$th basis vector in $n$ dimensions, namely a vector with a 1 in the $i$th coordinate, and 0's elsewhere. We shall use $\mathbf{1}[\cdot]$ to denote the "indicator function", where $\mathbf{1}[\texttt{predicate}]$ is 1 if $\texttt{predicate}$ is true, and 0 if it is false. It may be that $\texttt{predicate}$ is a random variable, in which case $\mathbf{1}[\texttt{predicate}]$ is a random variable as well.

### 2.1  The Setting: Budgeted Adversary Games

We will now describe the generic sequential decision problem, where a problem instance is characterized by the following triple: an $n \times n$ loss matrix $M$, a monotonic "cost function" cost : $[n]^* \to \mathbb{R}_+$, and a cost budget $k$. A cost function is *monotonic* as long as it satisfies the relation $\text{cost}(\rho\sigma) \leq \text{cost}(\rho i \sigma)$ for all $\rho, \sigma \in [n]^*$ and all $i \in [n]$. Play proceeds as follows:

1. On each round $t$, the player chooses a distribution $\mathbf{w}_t \in \Delta_n$ over his action space.

2. An outcome $i_t \in [n]$ is chosen by Nature (potentially an adversary).

3. The player suffers $\mathbf{w}_t^\top M \mathbf{e}_{i_t}$.

4. The game proceeds until the first round in which the budget is spent, i.e. the round $T$ when $\text{cost}(i_1 i_2 \ldots i_{T-1}) \leq k < \text{cost}(i_1 i_2 \ldots i_{T-1} i_T)$.

The goal of the Player is to choose each $\mathbf{w}_t$ in order to minimize the total cost of this repeated game on all sequences of outcomes. Note, importantly, that the player can *learn* from the past, and hence would like an efficiently computable function $\mathbf{w} : [n]^* \to \Delta_n$, where on round $t$ the player is given $\rho_{t-1} = (i_1 \ldots i_{t-1})$ and sets $\mathbf{w}_t \leftarrow \mathbf{w}(\rho_{t-1})$. We can define the worst-case cost of an algorithm

$\mathbf{w} : [n]^* \to \Delta_n$ by its performance against a worst-case sequence, that is

$$\text{WorstCaseLoss}(\mathbf{w}; M, \text{cost}, k) := \max_{\substack{\rho = i_1 i_2 \ldots \in [n]^* \\ \text{cost}(\rho_{T-1}) \le k < \text{cost}(\rho_T)}} \sum_{t=1}^{T} \mathbf{w}(\rho_{t-1})^\top M \mathbf{e}_{i_t}.$$

Note that above $T$ is a parameter chosen according to $\rho$ and the budget. We can also define the minimax loss, which is defined by choosing the $\mathbf{w}(\cdot)$ which minimizes $\text{WorstCaseLoss}(\cdot)$. Specifically,

$$\text{MinimaxLoss}(M, \text{cost}, k) := \min_{\mathbf{w}:[n]^* \to \Delta_n} \max_{\substack{\rho = i_1 i_2 \ldots \in [n]^* \\ \text{cost}(\rho_{T-1}) \le k < \text{cost}(\rho_T)}} \sum_{t=1}^{T} \mathbf{w}(\rho_{t-1})^\top M \mathbf{e}_{i_t}.$$

In the next section, we describe the optimal algorithm for a restricted class of $M$. That is, we obtain the mapping $\mathbf{w}$ which optimizes $\text{WorstCaseLoss}(\mathbf{w}, M, \text{cost}, k)$.

## 3 The Algorithm

We will start by assuming that $M$ is a nonnegative diagonal matrix, that is $M = \text{diag}(c_1, c_2, \ldots, c_n)$, and $c_i > 0$ for all $i$. With these values $c_i$, define the distribution $\mathbf{q} \in \Delta_n$ with $q_i := \frac{1/c_i}{\sum_j 1/c_j}$.

Given a current state $\rho$, the algorithm will rely heavily on our ability to compute the following function $\Phi(\cdot)$. For any $\rho \in [n]^*$ such that $\text{cost}(\rho) > k$, define $\Phi(\rho) := 0$. Otherwise, let

$$\Phi(\rho) := \frac{1}{\sum_i 1/c_i} \mathop{\mathbb{E}}_{\forall t: i_t \sim \mathbf{q}} \left[ \sum_{t=0}^{\infty} \mathbf{1}[\text{cost}(\rho i_1 \ldots i_t) \le k] \right]$$

Notice, this is the expected length of a random process. Of course, we must impose the natural condition that the length of this process has a finite expectation. Also, since we assume that the cost increases, it is reasonable to require that the distribution over the length, i.e. $\min\{t : \text{cost}(\rho i_1 \ldots i_t) > k\}$, has an exponentially decaying tail. Under these weak conditions, the following $m$-trial Monte Carlo method will provide a high probability estimate to error within $O(m^{-1/2})$.

---

**Algorithm 1** Efficient Estimation of $\Phi(\rho)$

**for** i=1...m **do**
    Sample: infinite random sequence $\sigma := i_1 i_2 \ldots$ where $\Pr(i_t = i) = q_i$
    Let: $T_i = \max\{t : \text{cost}(\rho \sigma_{t-1}) \le k\}$
**end for**
Return $\frac{\sum_{i=1}^{m} T_i}{m}$

---

Notice that the infinite sequence $\sigma$ does not have to be fully generated. Instead, we can continue to sample the sequence and simply stop when the condition $\text{cost}(\rho \sigma_{t-1}) \ge k$ is reached. We can now define our algorithm in terms of $\Phi(\cdot)$.

---

**Algorithm 2** Player's optimal strategy

Input: state $\rho$
Compute: $\Phi(\rho), \Phi(\rho, 1), \Phi(\rho, 2), \ldots, \Phi(\rho, n)$
Let: set $\mathbf{w}(\rho)$ with values $w_i(\rho) = \frac{\Phi(\rho) - \Phi(\rho, i)}{c_i}$

---

## 4 Minimax Optimality

Now we prove that Algorithm 2 is both "legal" and minimax optimal.

**Lemma 4.1.** *The vector $\mathbf{w}(\rho)$ computed in Algorithm 2 is always a valid distribution.*

*Proof.* It must first be established that $w_i(\rho) \geq 0$ for all $i$ and $\rho$. This, however, follows because we assume that the function cost() is monotonic, which implies that $\text{cost}(\rho\sigma) \leq \text{cost}(\rho i\sigma)$ and hence $\text{cost}(\rho i\sigma) \leq k \implies \text{cost}(\rho\sigma) \leq k$, and hence $\mathbf{1}[\text{cost}(\rho i\sigma) \leq k] \leq \mathbf{1}[\text{cost}(\rho\sigma) \leq k]$. Taking the expected difference of the infinite sum of these two indicators leads to $\Phi(\rho) - \Phi(\rho i) \geq 0$, which implies $w_i(\rho) \geq 0$ as desired.

We must also show that $\sum_i w_i(\rho) = 1$. We claim that the following recurrence relation holds for the function $\Phi(\rho)$ whenever $\text{cost}(\rho) \leq k$:

$$\Phi(\rho) = \underbrace{\frac{1}{\sum_i 1/c_i}}_{\text{first step}} + \underbrace{\sum_i q_i \Phi(\rho i)}_{\text{remaining steps}}, \text{ for any } \rho \text{ s.t. } \text{cost}(\rho) < k.$$

This is clear from noticing that $\Phi$ is an expected random walk length, with transition probabilities defined by $\mathbf{q}$, and scaled by the constant $(\sum_i 1/c_i)^{-1}$. Hence,

$$\sum_i w_i(\rho) = \sum_i \frac{\Phi(\rho) - \Phi(\rho i)}{c_i} = \left(\sum_i 1/c_i\right)\Phi(\rho) - \sum_i \frac{\Phi(\rho i)}{c_i}$$

$$= \left(\sum_i 1/c_i\right)\left(\frac{1}{\sum_i 1/c_i} + \sum_i q_i \Phi(\rho i)\right) - \sum_i \frac{\Phi(\rho i)}{c_i} = 1$$

where the last equality holds because $q_i = \frac{1/c_i}{\sum_j 1/c_j}$. $\quad\square$

**Theorem 4.1.** *For $M = diag(c_1, \ldots, c_n)$, Algorithm 2 is minimax optimal for the Budgeted Adversary problem. Furthermore, $\Phi(\emptyset) = \text{MinimaxLoss}(M, \text{cost}, k)$.*

*Proof.* First we prove an upper bound. Notice that, for an sequence $\rho = i_1 i_2 i_3 \ldots i_T$, the total cost of Algorithm 2 will be

$$\sum_{t=1}^T \mathbf{w}(\rho_{t-1})^\top M \mathbf{e}_{i_t} = \sum_{t=1}^T w_{i_t}(\rho_{t-1})c_{i_t} = \sum_{t=1}^T \frac{\Phi(\rho_{t-1}) - \Phi(\rho_t)}{c_{i_t}} c_{i_t} = \Phi(\emptyset) - \Phi(\rho_T) \leq \Phi(\emptyset)$$

and hence the total cost of the algorithm is always bounded by $\Phi(\emptyset)$.

On the other hand, we claim that $\Phi(\emptyset)$ can always be achieved by an adversary for any algorithm $\mathbf{w}'(\cdot)$. Construct a sequence $\rho$ as follows. Given that $\rho_{t-1}$ has been constructed so far, select any coordinate $i_t \in [n]$ for which $w_{i_t}(\rho_{t-1}) \leq w'_{i_t}(\rho_{t-1})$, that is, where the the algorithm $\mathbf{w}'$ places at least as much weight on $i_t$ as the proposed algorithm $\mathbf{w}$ we defined in Algorithm 2. This must always be possible because both $\mathbf{w}(\rho_{t-1})$ and $\mathbf{w}'(\rho_{t-1})$ are distributions and neither can fully dominate the other. Set $\rho_t \leftarrow \rho_{t-1}i$. Continue constructing $\rho$ until the budget is reached, i.e. $\text{cost}(\rho) > k$. Now, let us check the loss of $\mathbf{w}'$ on this sequence $\rho$:

$$\sum_{t=1}^T \mathbf{w}'(\rho_{t-1})^\top M \mathbf{e}_{i_t} = \sum_{t=1}^T w'_{i_t}(\rho_{t-1})c_{i_t} \geq \sum_{t=1}^T w_{i_t}(\rho_{t-1})c_{i_t} = \Phi(\emptyset) - \Phi(\rho) = \Phi(\emptyset)$$

Hence, an adversary can achieve at least $\Phi(\emptyset)$ loss for any algorithm $\mathbf{w}'$. $\quad\square$

## 4.1 Extensions

For simplicity of exposition, we proved Theorem 4.1 under a somewhat limited scope: only for diagonal matrices $M$, known budget $k$ and cost(). But with some work, these restrictions can be lifted. We sketch a few extensions of the result, although we omit the details due to lack of space.

First, the concept of a cost() function and a budget $k$ is not entirely necessary. Indeed, we can redefine the Budgeted Adversary game in terms of an arbitrary stopping criterion $\delta : [n]^* \to \{0, 1\}$, where $\delta(\rho) = 0$ is equivalent to "the budget has been exceeded". The only requirement is that $\delta()$ is monotonic, which is naturally defined as $\delta(\rho i\sigma) = 1 \implies \delta(\rho\sigma) = 1$ for all $\rho, \sigma \in [n]^*$ and all $i \in [n]$. This alternative budget interpretation lets us consider the sequence $\rho$ as a path through

a game tree. At a given node $\rho_t$ of the tree, the adversary's action $i_{t+1}$ determines which branch to follow. As soon as $\delta(\rho_t) = 0$ we have reached a terminal node of this tree.

Second, we need not assume that the budget $k$, or even the generalized stopping criterion $\delta()$, is known in advance. Instead, we can work with the following generalization: the stopping criterion $\delta$ is drawn from a known prior $\lambda$ and given to the adversary before the start of the game. The resulting optimal algorithm depends simply on estimating a new version of $\Phi(\rho)$. $\Phi(\rho)$ is now redefined as both an expectation over a random $\sigma$ and a random $\delta$ drawn from the *posterior* of $\lambda$, that is where we condition on the event $\delta(\rho) = 1$.

Third, Theorem 4.1 can be extended to a more general class of $M$, namely *inverse-nonnegative matrices*, where $M$ is invertible and $M^{-1}$ has all nonnegative entries. (In all the examples we give we need only diagonal $M$, but we sketch this generalization for completeness). If we let $\mathbf{1}_n$ be the vector of $n$ ones, then define $D = \mathrm{diag}^{-1}(M^{-1}\mathbf{1}_n)$, which is a nonnegative diagonal matrix. Also let $N = DM^{-1}$ and notice that the rows of $N$ are the normalized rows of $M^{-1}$. We can use Algorithm 2 with the diagonal matrix $D$, and attain distribution $\mathbf{w}'(\rho)$ for any $\rho$. To obtain an algorithm for the matrix $M$ (not $D$), we simply let $\mathbf{w}(\rho) = (\mathbf{w}'(\rho)^\top N)^\top$, which is guaranteed to be a distribution. The loss of $\mathbf{w}$ is identical to $\mathbf{w}'$ since $\mathbf{w}(\rho)^\top M = \mathbf{w}'(\rho)^\top D$ by construction.

Fourth, we have only discussed minimizing *loss* against a budgeted adversary. But all the results can be extended easily to the case where the player is instead maximizing gain (and the adversary is minimizing). A particularly surprising result is that the minimax strategy is *identical* in either case; that is, the the recursive definition of $w_i(\rho)$ is the same whether the player is maximizing or minimizing. However, the termination condition might change depending on whether we are minimizing or maximizing. For example in the expert setting, the game stops when all experts have cost larger than $k$ versus at least one expert has gain at least $k$. Therefore for the same budget size $k$, the minimax value of the gain version is typically smaller than the value of the loss version.

**Simplified Notation.** For many examples, including two that we consider below, recording the entire sequence $\rho$ is unnecessary—the only relevant information is the *number* of times each $i$ occurs in $\rho$ and not where it occurs. This is the case precisely when the function $\mathrm{cost}(\rho)$ is unchanged up to permutations of $\rho$. In such situations, we can consider a smaller state space, which records the "counts" of each $i$ in the sequence $\rho$. We will use the notation $\mathbf{s} \in \mathbb{N}^n$, where $\mathbf{s}_t = \mathbf{e}_{i_1} + \ldots + \mathbf{e}_{i_t}$ for the sequence $\rho_t = i_1 i_2 \ldots i_t$.

## 5 The Cost-Sensitive Hedge Setting

A straightforward application of Budgeted Adversary games is the "Hedge setting" introduced by Freund and Schapire [10], a version of the aforementioned experts setting. The minimax algorithm for this special case was already thoroughly developed by Abernethy et al [1]. We describe an interesting extension that can be achieved using our techniques which has not yet been solved.

The Hedge game goes as follows. A learner must predict a sequence of distributions $\mathbf{w}_t \in \Delta_n$, and receive a sequence of loss vectors $\ell_t \in \{0,1\}^n$. The total loss to the learner is $\sum_t \mathbf{w}_t \cdot \ell_t$, and the game ceases only once the best expert has more than $k$ errors, i.e. $\min_i \sum_t \ell_{t,i} > k$. The learner wants to minimize his total loss.

The natural way to transform the Hedge game into a Budgeted Adversary problem is as follows. We'll let $\mathbf{s}$ be the state, defined as the vector of cumulative losses of all the experts.

$$M = \begin{bmatrix} 1 & & \\ & \ddots & \\ & & 1 \end{bmatrix} \qquad \mathrm{cost}(\mathbf{s}) = \min_i s_i \qquad \sum_t \mathbf{w}_t \cdot \ell_t = \sum_t \mathbf{w}_t^\top M \mathbf{e}_{i_t}$$

The proposed reduction *almost* works, except for one key issue: this only allows cost vectors of the form $\ell_t = M\mathbf{e}_{i_t} = \mathbf{e}_{i_t}$, since by definition Nature chooses columns of $M$. However, as shown in Abernethy et al, this is not a problem.

**Lemma 5.1** (Lemma 11 and Theorem 12 of [1]). *In the Hedge game, the worst case adversary always chooses $\ell_t \in \{\mathbf{e}_1, \ldots, \mathbf{e}_n\}$.*

The standard and more well-known, although non-minimax, algorithm for the Hedge setting [10] uses a simple modification of the Weighted Majority Algorithm [14], and is described simply by

setting $w_i(\mathbf{s}) = \frac{\exp(-\eta s_i)}{\sum_j \exp(-\eta s_j)}$. With the appropriate tuning of $\eta$, it is possible to bound the total loss of this algorithm by $k + \sqrt{2k \ln n} + \ln n$, which is known to be roughly optimal in the limit. Abernethy et al [1] provide the minimax optimal algorithm, but state the bound in terms of an expected length of a random walk. This is essentially equivalent to our description of the minimax cost in terms of $\Phi(\emptyset)$.

A significant drawback of the Hedge result, however, is that it requires the losses to be uniformly bounded in $[0, 1]$, that is $\ell_t \in [0, 1]^n$. Ideally, we would like an algorithm and a bound that can handle non-uniform cost ranges, i.e. where expert $i$ suffers loss in some range $[0, c_i]$. The $\ell_{t,i} \in [0, 1]$ assumption is fundamental to the Hedge analysis, and we see no simple way of modifying it to achieve a tight bound. The simplest trick, which is just to take $c_{\max} := \max_i c_i$, leads to a bound of the form $k + \sqrt{2c_{\max} k \ln n} + c_{\max} \ln n$ which we know to be very loose. Intuitively, this is because only a single "risky" expert, with a large $c_i$, should not affect the bound significantly.

In our Budgeted Adversary framework, this case can be dealt with trivially: letting $M = \text{diag}(c_1, \dots, c_n)$ and $\text{cost}(\mathbf{s}) = \min_i s_i c_i$ gives us immediately an optimal algorithm that, by Theorem 4.1, we know to be minimax optimal. According to the same theorem, the minimax loss bound is simply $\Phi(\emptyset)$ which, unfortunately, is in terms of a random walk length. We do not know how to obtain a closed form estimate of this expression, and we leave this as an intriguing open question.

## 6 Metrical Task Systems

A classic problem from the Online Algorithms community is known as Metrical Task Systems (MTS), which we now describe. A player (decision-maker, algorithm, etc.) is presented with a finite metric space and on each of a sequence of rounds will occupy a single state (or point) within this metric space. At the beginning of each round the player is presented with a *cost vector*, describing the cost of occupying each point in the metric space. The player has the option to remain at the his present state and pay this states associated cost, or he can decide to switch to another point in the metric and pay the cost of the new state. In the latter case, however, the player must also pay the *switching cost* which is exactly the metric distance between the two points.

The MTS problem is a useful abstraction for a number of problems; among these is job-scheduling. An algorithm would like to determine on which machine, across a large network, it should process a job. At any given time point, the algorithm observes the number of available cycles on each machine, and can choose to migrate the job to another machine. Of course, if the subsequent machine is a great distance, then the algorithm also pays the travel time of the job migration through the network.

Notice that, were we given a sequence of cost vectors in advance, we could compute the optimal path of the algorithm that minimized total cost. Indeed, this is efficiently solved by dynamic programming, and we will refer to this as the *optimal offline cost*, or just the offline cost. What we would like is an algorithm that performs well relative to the offline cost without knowledge of the sequence of cost vectors. The standard measure of performance for an online algorithm is the *competitive ratio*, which is the ratio of cost of the online algorithm to the optimal offline cost. For all the results discussed below, we assume that the online algorithm can maintain a *randomized* state—a distribution over the metric—and pays the expected cost according to this random choice (Randomized algorithms tend to exhibit much better competitive ratios than deterministic algorithms).

When the metric is uniform, i.e. where all pairs of points are at unit distance, it is known that the competitive ratio is $O(\log n)$, where $n$ is the number of points in the metric; this was shown by Borodin, Linial and Saks who introduced the problem [4]. For general metric spaces, Bartal et al achieved a competitive ratio of $O(\log^6 n)$ [3], and this was improved to $O(\log^2 n)$ by Fiat and Mendel [9]. The latter two techniques, however, rely on a scheme of randomly approximating the metric space with a hierarchical tree metric, adding a (likely-unnecessary) multiplicative cost factor of $\log n$. It is widely believed that the minimax competitive ratio is $O(\log n)$ in general, but this gap has remained elusive for at least 10 years.

The most significant progress towards $O(\log n)$ is the 2007 work of Bansal et al [2] who achieved such a ratio for the case of "weighted-star metrics". A weighted star is a metric such that each point $i$ has a fixed distance $d_i$ from some "center state", and traveling between any state $i$ and $j$ requires

going through the center, hence incurring a switching cost of $d_i + d_j$. For weighted-star metrics, Bansal et al managed to justify two simplifications which are quite useful:

1. We can assume that the cost vector is of the form $\langle 0, \ldots, \infty, \ldots, 0 \rangle$; that is, all state receive 0 cost, except some state $i$ which receives an infinite cost.

2. When the online algorithm is currently maintaining a distribution $\mathbf{w}$ over the metric, and an infinite cost occurs at state $i$, we can assume[1] that algorithm incurs exactly $2d_i w_i$, exactly the cost of having $w_i$ probability weight enter and leave $i$ from the center.

Bansal et al provide an efficient algorithm for this setting using primal-dual techniques developed for solving linear programs. With the methods developed in the present paper, however, we can give the minimax optimal online algorithm under the above simplifications. Notice that the adversary is now choosing a sequence of states $i_1, i_2, i_3 \ldots \in [n]$ at which to assign an infinite cost. If we let $\rho = i_1 i_2 i_3 \ldots$, then the online algorithm's job is to choose a sequence of distributions $\mathbf{w}(\rho_t)$, and pays $2d_{i_{t+1}} w_{i_{t+1}}(\rho_t)$ at each step. In the end, the online algorithm's cost is compared to the offline MTS cost of $\rho$, which we will call $\text{cost}(\rho)$. Assume[2] we know the cost of the offline in advance, say it's $k$, and let us define $M = \text{diag}(2d_1, \ldots, 2d_n)$. Then the player's job is to select an algorithm $\mathbf{w}$ which minimizes

$$\max_{\substack{\rho = (i_1, \ldots, i_T) \\ \text{cost}(\rho) \le k}} \sum_{t=1}^{T} \mathbf{w}(\rho_{t-1})^\top M \mathbf{e}_{i_t}.$$

As we have shown, Algorithm 2 is minimax optimal for this setting. The competitive ratio of this algorithm is precisely $\limsup_{k \to \infty} \left( \frac{1}{k} \text{MinimaxLoss}(M, \text{cost}, k) \right)$. Notice the convenient trick here: by bounding a priori the cost of the offline at $k$, we can simply imagine playing this repeated game until the budget $k$ is achieved. Then the competitive ratio is just the worst-case loss over the offline cost, $k$. On the downside, we don't know of any easy way to bound the worst-case loss $\Phi(\emptyset)$.

# 7 Combinatorial Information Markets

We now consider the design of so-called cost-function-based information markets, a popular type of prediction market. This work is well-developed by Chen and Pennock [7], with much useful discussion by Chen and Vaughn [8]. We refer the reader to the latter work, which provides a very clear picture of the nice relationship between online learning and the design of information markets.

In the simplest setting, a prediction market is a mechanism for selling $n$ types of contract, where a contract of type $i$ corresponds to some potential future outcome, say "event $i$ will occur". The standard assumption is that the set of possible outcomes are mutually exclusive, so only one of the $n$ events will occur—for example, a pending election with $n$ competing candidates and one eventual winner. When a bettor purchases a contract of type $i$, the manager of the market, or "market maker", promises to pay out \$1 if the outcome is $i$ and \$0 otherwise.

A popular research question in recent years is how to design such prediction markets when the outcome has a combinatorial structure. An election might produce a complex outcome like a group of candidates winning, and a bettor may desire to bet on a complex predicate, such as "none of the winning candidates will be from my state". This question is explored in Hanson [13], although without much discussion of the relevant computational issues. The computational aspects of combinatorial information markets are addressed in Chen et al [6], who provide a particular hardness result regarding computation of certain price functions, as well as a positive result for an alternative type of combinatorial market. In the present section, we propose a new technique for designing combinatorial markets using the techniques laid out in the present work.

In this type of information market, the task of a market maker is to choose a price for each of the $n$ contracts, but where the prices may be set adaptively according to the present demand. Let $\mathbf{s} \in \mathbb{N}^n$ denote the current volume, where $s_i$ is the number of contracts sold of type $i$. In a cost-function-based market, these prices are set according to a given convex "cost function" $C(\mathbf{s})$ which

represents a potential on the demand. It is assumed that $C(\cdot)$ satisfies the relation $C(\mathbf{s} + \alpha\mathbf{1}) = C(\mathbf{s}) + \alpha$ for all $\mathbf{s}$, and $\alpha > 0$ and $\frac{\partial^2 C}{\partial s_i^2} > 0$. A typical example of such a cost function is $C(\mathbf{s}) = b \log \sum_{i=1}^{n} \exp(s_i/b)$ where $b$ is a parameter (see Chen and Pennock for further discussion [7]); it's easy to check this function satisfies the desired properties.

Given the current volume $\mathbf{s}$, the price of contract $i$ is set at $C(\mathbf{s} + \mathbf{e}_i) - C(\mathbf{s})$. This pricing scheme has the advantage that the total money earned in this market is easy to compute: it's *exactly* $C(\mathbf{s})$ regardless of the order in which the contracts were purchased. A disadvantage of this market, however, is that the posted prices (typically) sum to greater than \$1! A primary goal of an information market is to incentivize bettors to reveal their private knowledge of the outcome of an event. If a given bettor believes the true distribution of the outcome to be $\mathbf{q} \in \Delta_n$, he will have an incentive to purchase any contract $i$ for which the current price $p_i$ is smaller than $q_i$, thus providing positive expected reward (relative to his predicted distribution). Using this cost-function scheme, it is possible that $q_i < C(\mathbf{s} + \mathbf{e}_i) - C(\mathbf{s})$ for all $i$ and hence a bettor will have no incentive to bet.

We propose instead an alternative market mechanism that avoids this difficulty: for every given volume state $\mathbf{s}$, the market maker will advertise a price vector $\mathbf{w}(\mathbf{s}) \in \Delta_n$. If a contract of type $i$ is purchased, the state proceeds to $\mathbf{s} + \mathbf{e}_i$, and the market maker earns $w_i(\mathbf{s})$. If a sequence of contracts $i_1 i_2 \ldots$ is purchased, the market maker's total earning is $\sum_t \mathbf{w}(\mathbf{e}_{i_1} + \ldots + \mathbf{e}_{i_{t-1}}) \cdot \mathbf{e}_{i_t}$. On the other hand, if the final demand is $\mathbf{s}$, in the worst case the market maker may have to payout a total of $\max_i s_i$ dollars. If we assume the market maker has a fixed budget $k$ on the max number of contracts he is willing to sell, and wants to maximize the total earned money from selling contracts subject to this constraint, then we have[3] exactly a Budgeted Adversary problem: let $M$ be the identity and let $\text{cost}(\mathbf{s}) := \max_i s_i$.

This looks quite similar to the Budgeted Adversary reduction in the Hedge Setting described above, which is perhaps not too surprising given the strong connections discovered in Chen and Vaughn [8] between learning with experts and market design. But this reduction gives us additional power: we now have a natural way to design combinatorial prediction markets. We sketch one such example, but we note that many more can be worked out also.

Assume we are in a setting where we have $n$ election candidates, but some subset of size $m < n$ will become the "winners", and any such subset is possible. In this case, we can imagine a market maker selling a contract of type $i$ with the following promise: if candidate $i$ is in the winning subset, the payout is $1/m$ and 0 otherwise. For similar reasons as above, the market maker should sell contracts at prices $p_i$ where $\sum_i p_i = 1$. If we assume that market maker has a budget constraint of $k$ for the final payout, then we can handle this new setting within the Budgeted Adversary framework by simply modifying the cost function appropriately:

$$\text{cost}(\mathbf{s}) = \max_{U \subset [n], |U| = m} \sum_{i \in U} \frac{s_i}{m}.$$

This solution looks quite simple, so what did we gain? The benefit of our Budgeted Adversary framework is that we can handle *arbitrary* monotonic budget constraints, and the combinatorial nature of this problem can be encoded within the budget. We showed this for the case of "subset betting", but it can be applied to a wide range of settings with combinatorial outcomes.

# 8  Open problem

We have provided a very general framework for solving repeated zero-sum games against a budgeted adversary. Unfortunately, the generality of these results only go as far as games with payoff matrices that are inverse-nonnegative. For one-shot games, of course, Von Neumann's minimax theorem leads us to an efficient algorithm, i.e. linear programming, which can handle any payoff matrix, and we would hope this is achievable here. We thus pose the following open question: **Is there an efficient algorithm for solving Budgeted Adversary games for arbitrary matrices $M$?**

## Footnotes

[1] Precisely, they claim that it should be upper-bounded by $4d_i$. We omit the details regarding this issue, but it only contributes a multiplicative factor of 2 to the competitive ratio.

[2] Even when we do not know the offline cost in advance, standard "doubling tricks" allow you to guess this value and increase the guess as the game proceeds. For space, we omit these details.

[3]The careful reader may notice that this modified model may lead to a problem not present in the cost-function based markets: an arbitrage opportunity for the bettors. This issue can be dealt with by including a sufficient transaction fee per contract, but we omit these details due to space constraints.

# References

[1] J. Abernethy, M. K. Warmuth, and J. Yellin. Optimal strategies from random walks. In *Proceedings of the 21st Annual Conference on Learning Theory (COLT 08)*, pages 437–445, July 2008.

[2] Nikhil Bansal, Niv Buchbinder, and Joseph (Seffi) Naor. A Primal-Dual randomized algorithm for weighted paging. In *Proceedings of the 48th Annual IEEE Symposium on Foundations of Computer Science*, pages 507–517. IEEE Computer Society, 2007.

[3] Y. Bartal, A. Blum, C. Burch, and A. Tomkins. A polylog (n)-competitive algorithm for metrical task systems. In *Proceedings of the twenty-ninth annual ACM symposium on Theory of computing*, page 711719, 1997.

[4] A. Borodin, N. Linial, and M. E Saks. An optimal on-line algorithm for metrical task system. *Journal of the ACM (JACM)*, 39(4):745763, 1992.

[5] B. Brügmann. Monte carlo go. *Master's Thesis, Unpublished*, 1993.

[6] Y. Chen, L. Fortnow, N. Lambert, D. M Pennock, and J. Wortman. Complexity of combinatorial market makers. In *Proceedings of the ACM Conference on Electronic Commerce (EC)*, 2008.

[7] Y. Chen and D. M Pennock. A utility framework for bounded-loss market makers. In *Proceedings of the 23rd Conference on Uncertainty in Artificial Intelligence*, page 4956, 2007.

[8] Y. Chen and J. W Vaughan. A new understanding of prediction markets via No-Regret learning. *Arxiv preprint arXiv:1003.0034*, 2010.

[9] A. Fiat and M. Mendel. Better algorithms for unfair metrical task systems and applications. In *Proceedings of the thirty-second annual ACM symposium on Theory of computing*, page 725734, 2000.

[10] Yoav Freund and Robert E. Schapire. A decision-theoretic generalization of on-line learning and an application to Boosting. *J. Comput. Syst. Sci.*, 55(1):119–139, 1997. Special Issue for EuroCOLT '95.

[11] S. Gelly and D. Silver. Combining online and offline knowledge in UCT. In *Proceedings of the 24th international conference on Machine learning*, page 280, 2007.

[12] S. Gelly, Y. Wang, R. Munos, and O. Teytaud. Modification of UCT with patterns in Monte-Carlo go. 2006.

[13] R. Hanson. Combinatorial information market design. *Information Systems Frontiers*, 5(1):107119, 2003.

[14] N. Littlestone and M. K. Warmuth. The Weighted Majority algorithm. *Inform. Comput.*, 108(2):212–261, 1994. Preliminary version in FOCS 89.

